# Spoken Letter Recognition

**Mark Fanty & Ronald Cole**
Dept. of Computer Science and Engineering
Oregon Graduate Institute
Beaverton, OR 97006

## Abstract

Through the use of neural network classifiers and careful feature selection, we have achieved high-accuracy speaker-independent spoken letter recognition. For isolated letters, a broad-category segmentation is performed Location of segment boundaries allows us to measure features at specific locations in the signal such as vowel onset, where important information resides. Letter classification is performed with a feed-forward neural network. Recognition accuracy on a test set of 30 speakers was 96%. Neural network classifiers are also used for pitch tracking and broad-category segmentation of letter strings. Our research has been extended to recognition of names spelled with pauses between the letters. When searching a database of 50,000 names, we achieved 95% first choice name retrieval. Work has begun on a continuous letter classifier which does frame-by-frame phonetic classification of spoken letters.

## 1 INTRODUCTION

Although spoken letter recognition may seem like a modest goal because of the small vocabulary size, it is a most difficult task. Many letter pairs, such as M-N and B-D, differ by a single articulatory feature. Recent advances in classification technology have enabled us to achieve new levels of accuracy on this task [Cole *et al.*, 1990, Cole and Fanty, 1990, Fanty and Cole, 1990]. The EAR (English Alphabet Recognition) system developed in our laboratory recognizes letters of the English alphabet, spoken in isolation by any speaker, at 96% accuracy. We achieve this level of accuracy by training neural network classifiers with empirically derived features—features selected on the basis of speech knowledge, and refined through

experimentation. This process results in significantly better performance than just using "raw" data such as spectral coefficients.

We have extended our research to retrieval of names from spellings with brief pauses between the letters, and to continuous spellings. This paper provides an overview of these systems with an emphasis on our use of neural network classifiers for several separate components. In all cases, we use feedforward networks, with full connectivity between adjacent layers. The networks are trained using back propagation with conjugate gradient descent.

## 2  ISOLATED LETTER RECOGNITION

### 2.1  SYSTEM OVERVIEW

**Data capture** is performed using a Sennheiser HMD 224 noise-canceling microphone, lowpass filtered at 7.6 kHz and sampled at 16 kHz per second.

**Signal processing** routines produce the following representations every 3 msecs: (a) zero crossing rate: the number of zero crossings of the waveform in a 10 msec window; (b) amplitude: the peak-to-peak amplitude (largest positive value minus largest negative value) in a 10 msec window in the waveform; (c) filtered amplitude: the peak-to-peak amplitude in a 10 msec window in the waveform lowpass filtered at 700 Hz; (d) DFT: a 256 point FFT (128 real numbers) computed on a 10 msec Hanning window; and (e) spectral difference: the squared difference of the averaged spectra in adjacent 24 msec intervals.

**Pitch tracking** is performed with a neural network which locates peaks in the filtered (0-700 Hz) waveform that begin pitch periods. described in section 2.2.

**Broad-category segmention** divides the utterance into contiguous intervals and assigns one of four broad category labels to each interval: CLOS (closure or background noise), SON (sonorant interval), FRIC (fricative) and STOP. The segmenter, modified from [April 1988], uses cooperating knowledge sources which apply rules to the signal representations, most notably ptp0-700, pitch and zc0-8000.

**Feature measurement** is performed on selected locations in the utterance, based upon the broad-category boundaries. A total of 617 inputs are used by the classifier.

**Letter classification** is performed by a network with 52 hidden units and 26 output units, one per letter.

### 2.2  NEURAL NETWORK PITCH TRACKER

Pitch tracking is achieved through a network which classifies each peak in the waveform as to whether it begins a pitch period [Barnard *et al.*, 1991]. The waveform is lowpass filtered at 700 Hz and each positive peak is classified using information about it and the preceding and following four peaks. For each of the nine peaks, the following information is provided. (1) the amplitude, (2) the time difference between the peak and the candidate peak, (3) a measure of the similarity of the peak and the candidate peak (point-by-point correlation), (4) the width of the peak, and (5) the negative amplitude or most negative value preceding the peak. The network

was trained on the TIMIT database, and agrees with expert labelers about 98% of the time. It performs well on our data without retraining.

## 2.3   NEURAL NETWORK LETTER CLASSIFIER

Each letter (except W) has a single SON segment (e.g. the /iy/ in T, the whole letter M). This segment always exists, and provides the temporal anchor for most of the feature measurements. The previous consonant is the STOP or FRIC (e.g. B or C) before the SON. If there is no STOP or FRIC (e.g. E), the 200 msec interval before the SON is treated as a single segment for feature extraction. After dozens of experiments, we arrived at the following feature set:

- DFT coefficients from the consonant preceding the SON. The consonant is divided into thirds temporally; from each third, 32 averaged values are extracted linearly from 0 to 8kHz. All DFT inputs are normalized locally so that the largest value from a given time slice becomes 1.0 and the smallest becomes 0.0. (96 values)
- DFT coefficients from the SON. From each seventh of the SON, 32 averaged values are extracted linearly from 0 to 4kHz. (224 values)
- DFT coefficients following the SON. At the point of maximum zero-crossing rate in the 200 msec after the SON, 32 values are extracted linearly from 0 to 8kHz. (32 values)
- DFT coefficients from the second and fifth frame of the SON—32 values from each frame extracted linearly from 0 to 4kHz. These are not averaged over time, and will reflect formant movement at the SON onset. (64 values)
- DFT coefficients from the location in the center of the SON with the largest spectral difference—linear from 0 to 4kHz. This samples the formant locations at the vowel-nasal boundary in case the letter is M or N. (32 values)
- Zero-crossing rate in 11 18-msec segments (198 msec) before the SON, in 11 equal-length segments during the SON and in 11 18-msec segments after the SON. This provides an absolute time scale before and after the SON which could help overcome segmentation errors. (33 values)
- Amplitude from before, during and after the SON represented the same way as zero-crossing. (33 values)
- Filtered amplitude represented the same way as amplitude. (33 values)
- Spectral difference represented like zero-crossing and amplitude except the maximum value for each segment is used instead of the average, to avoid smoothing the peaks which occur at boundaries. (33 values)
- Inside the SON, the spectral center of mass from 0 to 1000 Hz, measured in 10 equal segments. (10 values)
- Inside the SON, the spectral center of mass from 1500 to 3500 Hz, measured in 10 equal segments. (10 values)
- Median pitch, the median distance between pitch peaks in the center of the SON. (1 value)
- Duration of the SON. (1 value)

- Duration of the consonant before the SON. (1 value)
- High-resolution representation of the amplitude at the SON onset: five values from 12 msec before the onset to 30 msec after the onset. (5 values)
- Abruptness of onset of the consonant before the SON, measured as the largest two-frame jump in amplitude in the 30 msec around the beginning of the consonant. (1 value)
- The label of the segment before the SON: CLOS, FRIC or STOP. (3 values)
- The largest spectral difference value from 100 msec before the SON onset to 21 msec after, normalized to accentuate the difference between B and V. (1 value)
- The number of consistent pitch peaks in the previous consonant. (1 value)
- The number of consistent pitch peaks before the previous consonant. (1 value)
- The presence of the segment sequence CLOS FRIC after the SON (an indicator of X or H). (1 binary value)

All inputs to our network were normalized: mapped to the interval [0.0, 1.0]. We attempted to normalize so that the entire range was well utilized. In some instances, the normalization was keyed to particular distinctions. For example, the center of mass in the spectrum from 0 to 1000 Hz was normalized so that E was low and A was high. Other vowels, such as O would have values "off the scale" and would map to 1.0, but the feature was added specifically for E/A distinctions.

## 2.4   PERFORMANCE

During feature development, two utterances of each letter from 60 speakers were used for training and 60 additional speakers served as the test set. For the final performance evaluation, these 120 speakers were combined to form a large training set. The final test set consists of 30 new speakers. The network correctly classified 95.9% of the letters.

The E-set {B,C,D,E,G,P,T,V,Z} and MN are the most difficult letters to classify. We trained separate network for just the M vs. N distinction and another for just the letters in the E-set [Fanty and Cole, 1990]. Using these networks as a second pass when the first network has a response in the E-set or in {M,N}, the performance rose slightly to 96%.

As mentioned above, all feature development was performed by training on half the training speakers and testing on the other half. The development set performance was 93.5% when using all the features. With only the 448 DFT values (not spectral difference or center of mass) the performance was 87%. Using all the features except DFT values (but including spectral difference and center of mass), the performance was 83%.

## 3   NAME RETRIEVAL FROM SPELLINGS

### 3.1   SYSTEM OVERVIEW

Our isolated letter recognizer was expanded to recognize letters spoken with pauses by (1) Training a neural network to do broad-category segmentation of spelled

strings (described in section 3.2); (2) Retraining the letter recognizer using letters extracted from spelled strings; (3) Devising an algorithm to divide an utterance into individual letters based on the broad category segmentation; and (4) Efficiently searching a large list of names to find the best match.

The letter classification network uses the same features as the isolated letter network. Feature measurements were based on segment boundaries provided by the neural network segmenter. The letter classification network was trained on isolated letters from 120 speakers plus letters from spelled strings from 60 additional speakers. The letter recognition performance on our cross-validation set of 8 speakers was 97%; on a preliminary test set of 10 additional speakers it was 95.5%. The letter recognition performance on our final test set was lower, as reported below.

The rules for letter segmentation are simplified by the structure of the English alphabet. All letters (except W—see below) have a single syllable, which corresponds to a single SON segment in the broad-category segmentation. In the usual case, letter boundaries are placed at the last CLOS or GLOT between SONs. A full description of the rules used can be found in [Cole *et al.*, 1991].

The output of the classifier is a score between 0.0 and 1.0 for each letter. These scores are treated as probabilities and the most likely name is retrieved from the database. The names are stored in a tree structure. The number of nodes near the root of the tree is small, so the search is fast. As the search approaches the leaves, the number of nodes grows rapidly, but it is possible to prune low-scoring paths.

## 3.2   NEURAL NETWORK BROAD-CATEGORY SEGMENTATION

The rule-based segmenter developed for isolated letters was too finely tuned to work well on letter strings. Rather than re-tune the rules, we decided to train a network to do broad category segmentation. At the same time, we added the category GLOT for glottalization, a slowing down of the vocal cords which often occurs at vowel-vowel boundaries.

The rule-based segmenter searched for boundaries. The neural network segmenter works in a different way [Gopalakrishnan, August 1990]. It classifies each 3 msec frame as being in a SON, CLOS, STOP, FRIC or GLOT. A five-point median smoothing is applied to the outputs, and the classification of the frame is taken to be the largest output. Some simple rules are applied to delete impossible segments such as 12 msec SONs.

The features found to produce the best performance are:

- 64 DFT coefficients linear from 0 to 8kHz at the frame to be classified.
- Spectral difference of adjacent 24 msec segments. These values are given for every frame in the 30 msec surrounding the frame to be classified, and for every 5 frames beyond that to 150 msecs before and after the frame to be classified. All subsequent features are sampled in the same manner.
- Spectral difference from 0 to 700 Hz in adjacent 24 msec segments.
- Amplitude of the waveform.
- Amplitude of the waveform lowpass filtered at 700 Hz. The window used to

measure the amplitude is just larger than the median pitch. In normal voicing, there is always at least one pitch peak inside the window and the output is smooth. During glottalization, the pitch peaks are more widely spaced. For some frames, the window used to measure amplitude contains no pitch peaks and the amplitude is sharply lower. Uneveness in this measure is thus an indication of glottalization.

- Zero crossing rate.
- A binary indicator of consistent pitch.
- The center of mass in the DFT coefficients between 0 and 1000 Hz.

A train-on-errors procedure was found to be very helpful. The segmenter resulting from training on the initial data set was used to classify new data. Frames for which it disagreed with hand-labeling were added to the initial data set and the network was retrained. This process was repeated several times.

## 3.3  SYSTEM PERFORMANCE

The system was evaluated on 1020 names provided by 34 speakers who were not used to train the system. Each subject spelled 30 names drawn randomly from the database of 50,000 surnames. The speaker was instructed to pause briefly between letters, but was not given any feedback during the session.

The list of 50,000 names provides a grammar of possible strings with perplexity 4. Using this grammar, the correct name was found 95.3% of the time. Of the 48 names not correctly retrieved, all but 6 of these were in the top 3 choices, and all but 2 were in the top 10. The letter recognition accuracy was 98.8% (total words minus substitutions plus deletions plus insertions, using a dynamic programming match). Examination of these name-retrieval errors revealed that about 50% were caused by misclassification of a letter, and about 50% were caused by bad letter segmentation. (Sixty percent of the segmentation errors were caused by GLOT insertions; forty percent were due to the speaker failing to pause.)

Without a grammar, the correct name is found only 53.9% of the time; almost half the inputs had at least one segmentation or classification error. The letter recognition accuracy was 89.1% using a dynamic programming match. Ignoring segmentation errors, 93% of the letters were correctly classified.

## 4  PHONEME RECOGNITION IN CONNECTED LETTERS

We have begun work on a continuous letter recognizer, which does not require pauses between the letters. The current system has two parts: a phonetic classifier which categorizes each frame as one of 30 phonemes (those phonemes found in letters plus glottalization)[Janssen et al., 1989], and a Viterbi search to find the sequence of letters which best matches the frame-by-frame phoneme scores.

The phonetic classifier is given 160 DFT coefficients; 40 for the frame to be classified, for the immediate context (+/- 12 msec), for the near context (+/- 45 msec) and for

the far context (+/- 78 msec) In addition, 87 features are added for the waveform amplitude, zero-crossing rate and spectral difference measure in a 183 msec window centered on the frame to be classified. It was trained on 9833 frames from 130 speakers spelling naturally. It was tested on 72 new speakers and achieved 76% frame accuracy with the instances of each phoneme categories equally balanced. When we feed the outputs of this network into a second network in addition to the DFT and other features, performance rose to 81%.

Simple letter models are used in a Viterbi search and enforce order and duration constraints for the phonemes. More work is required on coarticulation modeling, among other things. We are especially anxious to use carefully chosen features as with our isolated letter recognizer.

## Acknowledgements

This research was supported by Apple Computer, NSF, and a grant from DARPA to the Computer Science & Engineering Department of the Oregon Graduate Institute. We thank Vince Weatherill for his help in collecting and labeling data.

## References

[Barnard et al., 1991] E. Barnard, R. A. Cole, M. Vea, and F. Alleva. Pitch detection with a neural net classifier. *IEEE Transactions on Acoustics, Speech and Signal Processing)*, 1991. To appear.

[Cole and Fanty, 1990] R. A. Cole and M. Fanty. Spoken letter recognition. In *Proceedings of the DARPA Workshop on Speech and Natural Language Processing*, June 1990. Hidden Valley, PA.

[Cole and Hou, April 1988] R. A. Cole and L. Hou. Segmentation and broad classification of continuous speech. In *Proceedings IEEE International Conference on Acoustics, Speech, and Signal Processing*, April, 1988.

[Cole et al., 1990] R. A. Cole, M. Fanty, Y. Muthusamy, and M. Gopalakrishnan. Speaker-independent recognition of spoken english letters. In *Proceedings of the International Joint Conference on Neural Networks*, June 1990. San Diego, CA.

[Cole et al., 1991] R. A. Cole, M. Fanty, M. Gopalakrishnan, and R. Janssen. Speaker-independent name retrieval from spellings using a database of 50,000 names. In *Proceedings IEEE International Conference on Acoustics, Speech, and Signal Processing*, 1991. Toronto, Canada.

[Fanty and Cole, 1990] M. Fanty and R. A. Cole. Speaker-independent english alphabet recognition: Experiments with the e-set. In *Proceedings of the International Conference on Spoken Language Processing*, November 1990. Kobe, Japan.

[Gopalakrishnan, August 1990] M. Gopalakrishnan. Segmenting speech into broad phonetic categories using neural networks. Master's thesis, Oregon Graduate Institute / Dept. of Computer Science, August, 1990.

[Janssen et al., ] R. D. T. Janssen, M. Fanty, and R. A. Cole. Speaker-independent phonetic classification of the english alphabet. submitted to *Proceedings of the International Joint Conference on Neural Networks*, 1991.